# Effects of Spike Timing Underlying Binocular Integration and Rivalry in a Neural Model of Early Visual Cortex

**Erik D. Lumer**
Wellcome department of Cognitive Neurology
Institute of Neurology, University College of London
12 Queen Square, London, WC1N 3BG, UK

## Abstract

In normal vision, the inputs from the two eyes are integrated into a single percept. When dissimilar images are presented to the two eyes, however, perceptual integration gives way to alternation between monocular inputs, a phenomenon called binocular rivalry. Although recent evidence indicates that binocular rivalry involves a modulation of neuronal responses in extrastriate cortex, the basic mechanisms responsible for differential processing of conflicting and congruent stimuli remain unclear. Using a neural network that models the mammalian early visual system, I demonstrate here that the desynchronized firing of cortical-like neurons that first receive inputs from the two eyes results in rivalrous activity patterns at later stages in the visual pathway. By contrast, synchronization of firing among these cells prevents such competition. The temporal coordination of cortical activity and its effects on neural competition emerge naturally from the network connectivity and from its dynamics. These results suggest that input-related differences in relative spike timing at an early stage of visual processing may give rise to the phenomena both of perceptual integration and rivalry in binocular vision.

## 1  Introduction

The neural determinants of visual perception can be probed by subjecting the visual system to ambiguous viewing conditions - stimulus configurations that admit more

than one perceptual interpretation. For example, when a left-tilted grating is shown to the left eye and a right-tilted grating to the right eye, the two stimuli are momentarily perceived together as a plaid pattern, but soon only one line grating becomes visible, while the other is suppressed. This phenomenon, known as binocular rivalry, has long been thought to involve competition between monocular neurons within the primary visual cortex (V1), leading to the suppression of information from one eye (Lehky, 1988; Blake, 1989). It has recently been shown, however, that neurons whose activity covaries with perception during rivalry are found mainly in higher cortical areas and respond to inputs from both eyes, thus suggesting that rivalry arises instead through competition between alternative stimulus interpretations in extrastriate cortex (Leopold and Logothetis, 1996). Because eye-specific information appears to be lost at this stage, it remains unclear how the stimulus conditions (i.e. conflicting monocular stimuli) yielding binocular rivalry are distinguished from the conditions (i.e. matched monocular inputs) that produce stable single vision.

I propose here that the degree of similarity between the images presented to the two eyes is registered by the temporal coordination of neuronal activity in V1, and that changes in relative spike timing within this area can instigate the differential responses in higher cortical areas to conflicting or congruent visual stimuli. Stimulus and eye-specific synchronous activity has been described previously both in the lateral geniculate nucleus (LGN) and in the striate cortex (Gray et al., 1989; Sillito et al., 1994; Neuenschwander and Singer, 1996). It has been suggested that such synchrony may serve to bind together spatially distributed neural events into coherent representations (Milner, 1974; von der Malsburg, 1981; Singer, 1993). In addition, reduced synchronization of striate cortical responses in strabismic cats has been correlated with their perceptual inability to combine signals from the two eyes or to incorporate signals from an amblyopic eye (König et al., 1993; Roelfsema et al., 1994). However, the specific influences of interocular input-similarity on spike coordination in the striate cortex, and of spike coordination on competition in other cortical areas, remain unclear.

To examine these influences, a simplified neural model of an early visual pathway is simulated. In what follows, I first describe the anatomical and physiological constraints incorporated in the model, and then show that a temporal patterning of neuronal activity in its primary cortical area emerges naturally. By manipulating the relative spike timing of neuronal discharges in this area, I demonstrate its role in inducing differential responses in higher visual areas to conflicting or congruent visual stimulation. Finally, I discuss possible implications of these results for understanding the neural basis of normal and ambiguous perception in vivo.

## 2   Model overview

The model has four stages based on the organization of the mammalian visual pathway (Gilbert, 1993). These stages represent: (i) sectors of an ipsilateral ('left eye') and a contralateral ('right eye') lamina of the LGN, which relay visual inputs to the cortex; (ii) two corresponding monocular regions in layer 4 of V1 with different ocular dominance; (iii) a primary cortical sector in which the monocular inputs are first combined (called Vp in the model); and (iv) a secondary visual area of cortex in which higher-order features are extracted (Vs in the model; Fig. 1). Each stage consists of 'standard' integrate-and-fire neurons that are incorporated in synaptic networks. At the cortical stages, these units are grouped in local recurrent circuits that are similar to those used in previous modeling studies (Douglas et al., 1995; Somers et al., 1995). Synaptic interactions in these circuits are both excitatory and inhibitory between cells with similar orientation selectivity, but are restricted to in-

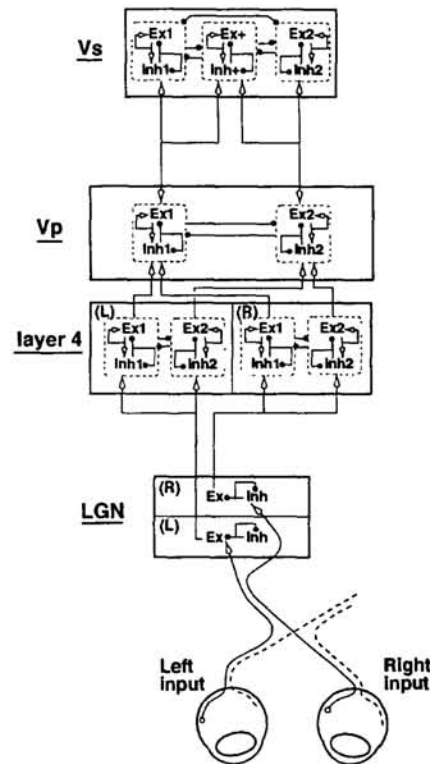

Figure 1: Architecture of the model. Excitatory and inhibitory connections are represented by lines with arrowheads and round terminals, respectively. Each lamina in the LGN consists of 100 excitatory units (Ex) and 100 inhibitory units (Inh), coupled via local inhibition. Cortical units are grouped into local recurrent circuits (stippled boxes), each comprising 200 Ex units and 100 Inh units. In each monocular patch of layer 4, one cell group (Ex1 and Inh1) responds to left-tilted lines (orientation 1), whereas a second group (Ex2 and Inh2) is selective for right-tilted lines (orientation 2). The same orientation selectivities are remapped onto Vp and Vs, although cells in these areas respond to inputs from both eyes. In addition, convergent inputs from Vp to Vs establish a third selectivity in Vs, namely for line crossings (Ex+ and Inh+).

hibition only between cell groups with orthogonal orientation preference (Kisvárday and Eysel, 1993). Two orthogonal orientations (orientation 1 and 2) are mapped in each monocular sector of layer 4, and in Vp. To account for the emergence of more complex response properties at higher levels in the visual system (Van Essen and Gallant, 1994), forward connectivity patterns from Vp to Vs are organized to support three feature selectivities in Vs, one for orientation 1, one for orientation 2, and one for the conjunction of these two orientations, i.e. for line crossings. These forward projections are reciprocated by weaker backward projections from Vs to Vp. As a general rule, connections are established at random within and between interconnected populations of cells, with connection probabilities between pairs of cells ranging from 1 to 10 %, consistent with experimental estimates (Thomson et al., 1988; Mason et al., 1991). Visual stimulation is achieved by applying a stochastic synaptic excitation independently to activated cells in the LGN. A quantitative description of the model parameters will be reported elsewhere.

# 3   Results

In a first series of simulations, the responses of the model to conflicting and congruent visual stimuli are compared. When the left input consists of left-tilted lines (orientation 1) and the right input of right-tilted lines (orientation 2), rivalrous response suppression occurs in the secondary visual area. At any moment, only one of the three feature-selective cell groups in Vs can maintain elevated firing rates (Fig. 2a). By contrast, when congruent plaid patterns are used to stimulate the two monocular channels, these cell groups are forced in a regime in which they all sustain elevated firing rates (Fig. 2b). This concurrent activation of cells selective for orthogonal orientations and for line crossings can be interpreted as a distributed representation of the plaid pattern in Vs [1]. A quantitative assessment of the degree of competition in Vs is shown in Figure 2c. The rivalry index of two groups of neurons is defined as the mean absolute value of the difference between their instantaneous group-averaged firing rates divided by the highest instantaneous firing rate among the two cell groups. This index varies between 0 for nonrivalrous groups of neurons and 1 for groups of neurons with mutually exclusive patterns of activity. Groups of cells with different selectivity in Vs have a significantly higher rivalry index when stimulated by conflicting rather than by congruent visual inputs ($p < 0.0001$) (Fig. 2c).

Note that, in the example shown in Figure 2a, the differential responses to conflicting inputs develop from about 200 ms after stimulus onset and are maintained over the remainder of the stimulation epoch. In other simulations, alternation between dominant and suppressed responses was also observed over the same epoch as a result of fluctuations in the overall network dynamics. A detailed analysis of the dynamics of perceptual alternation during rivalry, however, is beyond the scope of this report.

Although Vp exhibits a similar distribution of firing rates during rivalrous and nonrivalrous stimulation, synchronization between the two cell groups in Vp is more pronounced in the nonrivalrous than in the rivalrous case (Fig. 2d, upper plots). Subtraction of the shift predictor demonstrates that the units are not phase-locked to the stimuli. The changes in spike coordination among Vp units reflects the temporal patterning of their layer 4 inputs. During rivalry, Vp cells with different orientation selectivity are driven by layer 4 units that belong to separate monocular pathways, and hence, are uncorrelated (Fig. 2d, lower left). By contrast, cells in Vp receive convergent inputs from both eyes during nonrivalrous stimulation. Because of the synchronization of discharges among cells responsive to the same eye within layer 4 (Fig. 2d, lower right), the paired activities from the two monocular channels are also synchronized, and provide synchronous inputs to cells with different orientation selectivity in Vp.

To establish unequivocally that changes in spike coordination within Vp are sufficient to trigger differential responses in Vs to conflicting and congruent stimuli, the model can be modified as follows. A single group of cells in layer 4 is used to drive with equal strength both orientation-selective populations of neurons in Vp. The outputs from layer 4, however, is relayed to these two target populations with average transmission delays that differ by either 10 ms or by 0 ms. In the first case, competition prevails among cells in the secondary visual area. This contrasts with the nonrivalrous activity in this area when similar transmission delays are used at an earlier stage (data not shown). This test confirms that changes in relative spike

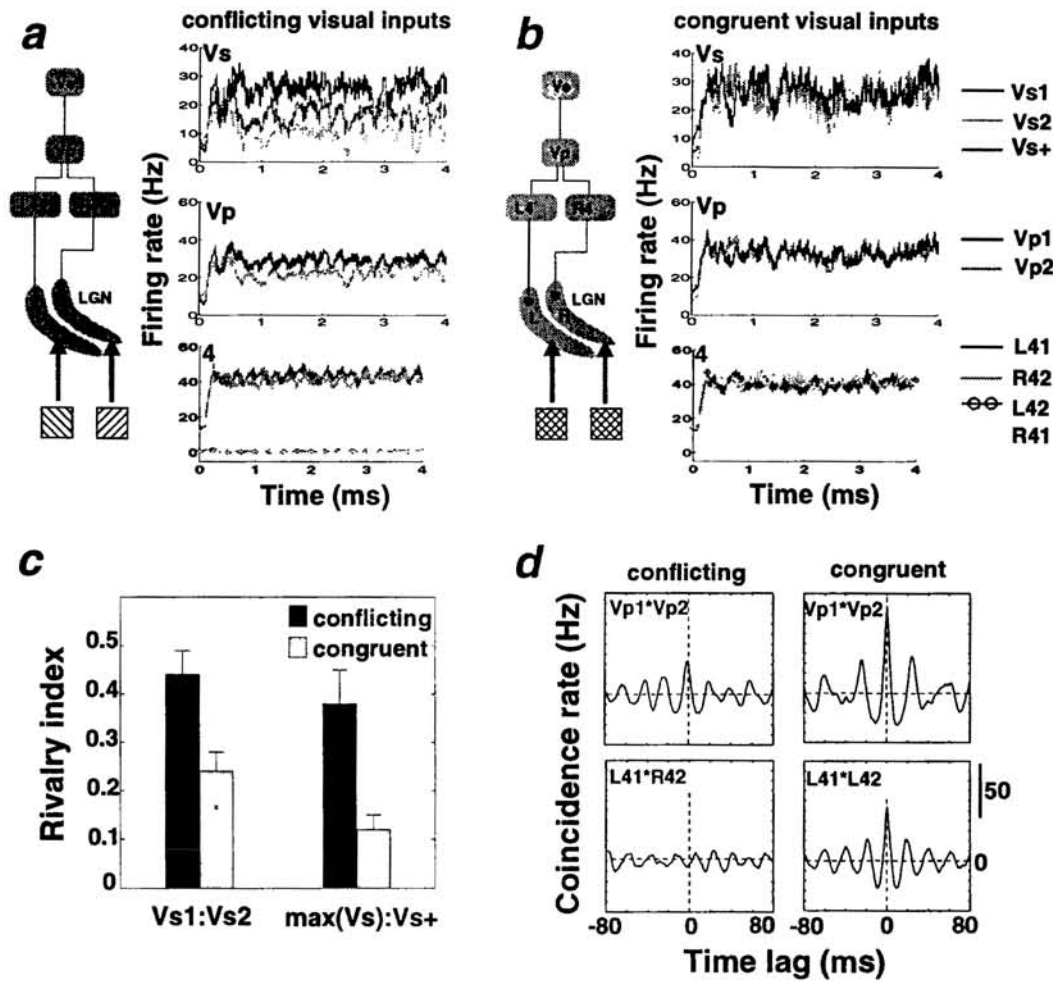

Figure 2: **A**, Instantaneous firing rates in response to conflicting inputs for cell groups in layer 4, in Vp, and in Vs (stimulus onset at $t = 250ms$). Discharge rates of layer 4 cells driven by different 'eyes' are similar (lower plot). By contrast, Vs exhibits competitive firing patterns soon after stimulus onset (upper plot). Feedback influence from Vs to Vp results in comparatively weaker competition in Vp (middle plot). **B**, Responses to congruent inputs. All cell groups in layer 4 are activated by visual inputs. Nonrivalrous firing patterns ensue in Vp and Vs. **C**, Rivalry indices during conflicting and congruent stimulation, are calculated for the two orientation-selective cell groups and for the dominant and cross-selective cell group in Vs. **D**, Interocular responses are uncorrelated in layer 4 (lower left), whereas intraocular activities are synchronous at this stage (lower right). Enhanced synchronization of discharges ensues between cell groups in Vp during congruent stimulation (upper right), relative to the degree of coherence during conflicting stimulation (upper left).

timing are sufficient to switch the outcome of neural network interactions involving strong mutual inhibition from competitive to cooperative.

# 4 Conclusion

In the present study, a simplified model of a visual pathway was used to gain insight into the neural mechanisms operating during binocular vision. Simulations of neuronal responses to visual inputs revealed a stimulus-related patterning of relative spike timing at an early stage of cortical processing. This patterning reflected the degree of similarity between the images presented to the two 'eyes', and, in turn, it altered the outcome of competitive interactions at later stages along the visual pathway. These effects can help explaining how the same cortical networks can exhibit both rivalrous and nonrivalrous activity, depending on the temporal coordination of their synaptic inputs.

These results bear on the interpretation of recent empirical findings about the neuronal correlates of rivalrous perception. In experiments with awake monkeys, Logothetis and colleagues (Sheinberg et al., 1995; Leopold and Logothetis, 1996) have shown that neurons whose firing rate correlates with perception during rivalry are distributed at several levels along the primate visual pathways, including V1/V2, V4, and IT. Importantly, the fraction of modulated responses is lower in V1 than in extrastriate areas, and it increases with the level in the visual hierarchy. Simulations of the present model exhibit a behavior that is consistent with these observations. However, these simulations also predict that both rivalrous and nonrivalrous perception may have a clear neurophysiological correlate in V1, i.e. at the earliest stage of visual cortical processing. Accordingly, congruent stimulation of both eyes will synchronize the firing of binocular cells with overlapping receptive fields in V1. By contrast, conflicting inputs to the two eyes will cause a desynchronization between their corresponding neural events in V1. Because this temporal registration of stimulus dissimilarity instigates competition among binocular cells in higher visual areas and not between monocular pathways, the ensuing pattern of response suppression and dominance is independent of the eyes through which the stimuli are presented. Thus, the model can in principle account for the psychophysical finding that a single phase of perceptual dominance during rivalry can span multiple interocular exchanges of the rival stimuli (Logothetis et al., 1996).

The present results also reveal a novel property of canonical cortical-like circuits interacting through mutual inhibition, i.e. the degree of competition among such circuits exhibits a remarkable sensitivity to the relative timing of neuronal action potentials. This suggests that the temporal patterning of cortical activity may be a fundamental mechanism for selecting among stimuli competing for the control of attention and motor action.

**Acknowledgements**

This work was supported in part by an IRSIA visiting fellowship at the Center for Nonlinear Phenomena and Complex Systems, Université Libre de Bruxelles. I thank Professor Grégoire Nicolis for his hospitality during my stay in Brussels; and David Leopold and Daniele Piomelli for helpful discussions and comments on an earlier version of the manuscript.

**References**

Blake R (1989) A neural theory of binocular vision. Psychol Rev 96:145-167.

Douglas RJ, Koch C, Mahowald M, Martin K, Suarez H (1995) Recurrent excitation in neocortical circuits. Science 269:981-985.

Gilbert C (1993) Circuitry, architecture, and functional dynamics of visual cortex. Cereb Cortex 3:373-386.

Gray CM, König P, Engel AK, Singer, W (1989) Oscillatory responses in cat visual cortex exhibit inter-columnar synchronization which reflects global stimulus properties. Nature 338:334-337.

Kisvárday ZF, Eysel UT (1993) Functional and structural topography of horizontal inhibitory connections in cat visual cortex. Europ J Neurosci 5:1558-1572.

König P, Engel AK, Löwel S, Singer, W (1993) Squint affects synchronization of oscillatory responses in cat visual cortex. Eur J Neurosci 5:501-508.

Lehky SR (1988) An astable multivibrator model of binocular rivalry. Perception 17: 215- 228.

Leopold DA, Logothetis NK (1996) Activity changes in early visual cortex reflect monkeys percepts during binocular rivalry. Nature 379:549-553.

Logothetis NK, Leopold DA, Sheinberg DL (1996) What is rivalling during rivalry? Nature 380:621-624.

Neuenschwander S, Singer W (1996) Long-range synchronization of oscillatory light responses in the cat retina and lateral geniculate nucleus. Nature 379:728-733.

Milner PM (1974) A model of visual shape recognition. Psychol Rev 81:521-535.

Roelfsema PR, König P, Engel AK, Sireteanu R, Singer W (1994) Reduced synchronization in the visual cortex of cats with strabismic amblyopia. Eur J Neurosci 6:1645-1655.

Sheinberg DL, Leopold DA, Logothetis NK (1995) Effects of binocular rivalry on face cell activity in monkey temporal cortex. Soc Neurosci Abstr 21:15.12.

Sillito AM, Jones HE, Gerstein GL, West DC (1994) Feature-linked synchronization of thalamic relay cell firing induced by feedback from the visual cortex. Nature 369:479-482.

Singer W (1993) Synchronization of cortical activity and its putative role in information processing. Annu Rev Physiol 55:349-374.

Somers D, Nelson S, Sur M (1995) An emergent model of orientation selectivity in cat visual cortical simple cells. J Neurosci 15:5448-5465.

Van Essen DC, Gallant JL (1994) Neural mechanisms of form and motion processing in the primate visual system. Neuron 13:1-10.

von der Malsburg C (1981) The correlation theory of the brain. Internal Report 81-2, Max Planck Institute for Biophysical Chemistry, Göttingen.

## Footnotes

[1]To discount possible effects of binocular summmation, synaptic strengths from layer 4 to Vp are reduced during congruent stimulation so as to produce a feedforward activation of Vp comparable to that elicited by conflicting monocular inputs.
